# Neurally Plausible Reinforcement Learning of Working Memory Tasks

**Jaldert O. Rombouts, Sander M. Bohte**
CWI, Life Sciences
Amsterdam, The Netherlands
{j.o.rombouts, s.m.bohte}@cwi.nl

**Pieter R. Roelfsema**
Netherlands Institute for Neuroscience
Amsterdam, The Netherlands
p.r.roelfsema@nin.knaw.nl

## Abstract

A key function of brains is undoubtedly the abstraction and maintenance of information from the environment for later use. Neurons in association cortex play an important role in this process: by learning these neurons become tuned to relevant features and represent the information that is required later as a persistent elevation of their activity [1]. It is however not well known how such neurons acquire these task-relevant working memories. Here we introduce a biologically plausible learning scheme grounded in Reinforcement Learning (RL) theory [2] that explains how neurons become selective for relevant information by trial and error learning. The model has memory units which learn useful internal state representations to solve working memory tasks by transforming partially observable Markov decision problems (POMDP) into MDPs. We propose that synaptic plasticity is guided by a combination of attentional feedback signals from the action selection stage to earlier processing levels and a globally released neuromodulatory signal. Feedback signals interact with feedforward signals to form synaptic tags at those connections that are responsible for the stimulus-response mapping. The neuromodulatory signal interacts with tagged synapses to determine the sign and strength of plasticity. The learning scheme is generic because it can train networks in different tasks, simply by varying inputs and rewards. It explains how neurons in association cortex learn to 1) temporarily store task-relevant information in non-linear stimulus-response mapping tasks [1, 3, 4] and 2) learn to optimally integrate probabilistic evidence for perceptual decision making [5, 6].

## 1 Introduction

By giving reward at the right times, animals like monkeys can be trained to perform complex tasks that require the mapping of sensory stimuli onto responses, the storage of information in working memory and the integration of uncertain sensory evidence. While significant progress has been made in reinforcement learning theory [2, 7, 8, 9], a generic learning rule for neural networks that is biologically plausible and also accounts for the versatility of animal learning has yet to be described.

We propose a simple biologically plausible neural network model that can solve a variety of working memory tasks. The network predicts action-values ($Q$-values) for different possible actions [2], and it learns to minimize SARSA [10, 2] temporal difference (TD) prediction errors by stochastic gradient descent. The model has memory units inspired by neurons in lateral intraparietal (LIP) cortex and prefrontal cortex. Such neurons exhibit persistent activations for task related cues in visual working memory tasks [1, 11, 4]. Memory units learn to represent an internal state that allows the network to solve working memory tasks by transforming POMDPs into MDPs [25]. The updates for synaptic weights have two components. The first is a synaptic tag [12] that arises from an interaction between feedforward and feedback activations. Tags form on those synapses that are responsible for the chosen actions by an attentional feedback process [13]. The second factor is a

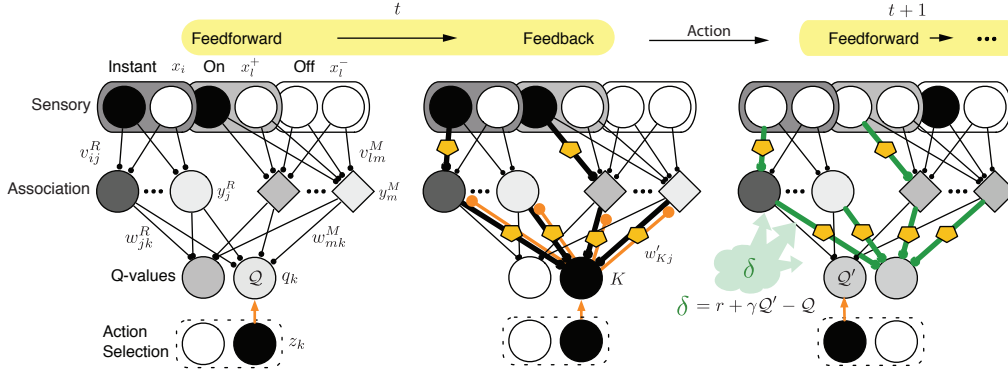

Figure 1: Model and learning (see section 2). Pentagons represent synaptic tags.

global neuromodulatory signal $\delta$ that reflects the TD error, and this signal interacts with the tags to yield synaptic plasticity. TD-errors are represented by dopamine neurons in the ventral tegmental area and substantia nigra [9, 14]. The persistence of tags permits learning if time passes between synaptic activity and the animal's choice, for example if information is stored in working memory or evidence accumulates before a decision is made. The learning rules are biologically plausible because the information required for computing the synaptic updates is available at the synapse. We call the new learning scheme AuGMEnT (Attention-Gated MEmory Tagging).

We first discuss the model and then show that it explains how neurons in association cortex learn to 1) temporarily store task-relevant information in non-linear stimulus-response mapping tasks [1, 3, 4] and 2) learn to optimally integrate probabilistic evidence for perceptual decision making [5, 6].

## 2 Model

AuGMEnT is modeled as a three layer neural network (Fig. 1). Units in the motor (output) layer predict $\mathcal{Q}$-values [2] for their associated actions. Predictions are learned by stochastic gradient descent on prediction errors.

The sensory layer contains two types of units; instantaneous and transient on(+)/off(-) units. Instantaneous units $x_i$ encode sensory inputs $s_i(t)$, and + and - units encode positive and negative changes in sensory inputs with respect to the previous time step $t - 1$:

$$x_i^+(t) = [s_i(t) - s_i(t-1)]_+ \; ; \qquad x_i^-(t) = [s_i(t-1) - s_i(t)]_+ \; , \tag{1}$$

where $[.]_+$ is a threshold operator that returns 0 for all negative inputs but leaves positive inputs unchanged. Each sensory variable $s_i$ is thus represented by three units $x_i, x_i^+, x_i^-$ (we only explicitly write the time dependence if it is ambiguous). We denote the set of differentiating units as $x'$. The hidden layer models the association cortex and it contains regular units and memory units. The regular units $j$ (Fig. 1, circles) are fully connected to the instantaneous units $i$ in the sensory layer by connections $v_{ij}^R$; $v_{0j}^R$ is a bias weight. Regular unit activations $y_j^R$ are computed as:

$$y_j^R = \sigma(a_j^R) = \frac{1}{1 + \exp\left(\theta - a_j^R\right)} \;\; \text{with} \;\; a_j^R = \sum_i v_{ij}^R x_i \; . \tag{2}$$

Memory units $m$ (Fig. 1, diamonds) are fully connected to the +/- units in the sensory layer by connections $v_{lm}^M$ and they derive their activations $y_j^M(t)$ by integrating their inputs:

$$y_m^M = \sigma(a_m^M) \;\; \text{with} \;\; a_m^M = a_m^M(t-1) + \sum_l v_{lm}^M x_l' \; , \tag{3}$$

with $\sigma$ as defined in eqn. (2). Output layer units $k$ are fully connected to the hidden layer by connections $w_{jk}^R$ (for regular hiddens, $w_{0k}^R$ is a bias weight) and $w_{mk}^M$ (for memory hiddens). Activations are computed as:

$$q_k = \sum_j y_j^R w_{jk}^R + \sum_m y_m^M w_{mk}^M \; . \tag{4}$$

A Winner Takes All (WTA) competition now selects an action based on the estimated $\mathcal{Q}$-values. We used a max-Boltzmann [15] controller which executes the action with the highest estimated $\mathcal{Q}$-value with probability $1 - \epsilon$ and otherwise it chooses an action with probabilities according to the Boltzmann distribution:

$$Pr(z_k = 1) = \frac{\exp q_k}{\sum_{k'} \exp q_{k'}} \ . \tag{5}$$

The WTA mechanism then sets the activation of the winning unit to 1 and the activation of all other units to 0; $z_k = \delta_{kK}$ where $\delta_{kK}$ is the Kronecker delta function. The winning unit sends feedback signals to the earlier processing layers, informing the rest of the network about the action that was taken. This feedback signal interacts with the feedforward activations to give rise to synaptic tags on those synapses that were involved in taking the decision. The tags then interact with a neuromodulatory signal $\delta$, which codes a TD error, to modify synaptic strengths.

## 2.1 Learning

After executing an action, the environment returns a new observation $\mathbf{s}'$, a scalar reward $r$, and possibly a signal indicating the end of a trial. The network computes a SARSA TD error [10, 2]:

$$\delta = r + \gamma q_{K'} - q_K \ , \tag{6}$$

where $q_{K'}$ is the predicted value of the winning action for the new observation, and $\gamma \in [0, 1]$ is the temporal discount parameter [2]. AuGMEnT learns by minimizing the squared prediction error $E$:

$$E = \frac{1}{2}(\delta)^2 = \frac{1}{2}(r + \gamma q_{K'} - q_K)^2 \ , \tag{7}$$

The synaptic updates have two factors. The first is a synaptic tag (Fig. 2, pentagons; equivalent to an eligibility trace in RL [2]) that arises from an interaction between feedforward and feedback activations. The second is a global neuromodulatory signal $\delta$ which interacts with these tags to yield synaptic plasticity. The updates can be derived by the chain rule for derivatives [16].

The update for synapses $w_{jk}^R$ is:

$$\Delta w_{jk}^R \ = \ -\beta \frac{\partial E}{\partial q_K} Tag_{jk}^R = \beta \delta(t) Tag_{jk}^R \ , \tag{8}$$

$$\Delta Tag_{jk}^R \ = \ (\lambda \gamma - 1) Tag_{jk}^R + \frac{\partial q_K}{\partial w_{jk}^R} = (\lambda \gamma - 1) Tag_{jk}^R + y_j^R z_k \ , \tag{9}$$

where $\beta$ is a learning rate, $Tag_{jk}^R$ are the synaptic tags on synapses between regular hidden units and the motor layer, and $\lambda$ is a decay parameter [2]. Note that $\Delta w_{jk}^R \propto -\beta \frac{\partial E}{\partial q_K} \frac{\partial q_K}{\partial w_{jk}^R} = -\beta \frac{\partial E}{\partial w_{jk}^R}$, holding with equality if $\lambda \gamma = 0$. If $\lambda \gamma > 0$, tags decay exponentially so that synapses that were responsible for previous actions are also assigned credit for the currently observed error.

Equivalently, updates for synapses between memory units and motor units are:

$$\Delta w_{mk}^M \ = \ \beta \delta(t) Tag_{mk}^M \ , \tag{10}$$

$$\Delta Tag_{mk}^M \ = \ (\lambda \gamma - 1) Tag_{mk}^M + y_m^M z_k \ . \tag{11}$$

The updates for synapses between instantaneous sensory units and regular association units are:

$$\Delta v_{ij}^R \ = \ -\beta \frac{\partial E}{\partial q_K} Tag_{ij}^R = \beta \delta Tag_{ij}^R \ , \tag{12}$$

$$\Delta Tag_{ij}^R \ = \ (\lambda \gamma - 1) Tag_{ij}^R + \frac{\partial q_K}{\partial y_j^R} \frac{\partial y_j^R}{\partial a_j^R} \frac{\partial a_j^R}{\partial v_{ij}^R} \ , \tag{13}$$

$$= \ (\lambda \gamma - 1) Tag_{ij}^R + w_{Kj}^{'R} y_j^R (1 - y_j^R) x_i \ , \tag{14}$$

where $w_{Kj}^{'R}$ are feedback weights from the motor layer back to the association layer. The intuition for the last equation is that the winning output unit $K$ provides feedback to the units in the association layer that were responsible for its activation. Association units with a strong feedforward connection also have a strong feedback connection. As a result, synapses onto association units that

provided strong input to the winning unit will have the strongest plasticity. This 'attentional feedback' mechanism was introduced in [13]. For convenience, we have assumed that feedforward and feedback weights are symmetrical, but they can also be trained as in [13].

For the updates for the synapses between +/- sensory units and memory units we first approximate the activation $a_m^M$ (see eqn. (3)) as:

$$a_m^M = a_m^M(t-1) + \sum_l v_{lm}^M x_l' \approx v_{lm}^M \sum_{t'=0}^{t} x_l'(t') , \tag{15}$$

which is a good approximation if the synapses $v_{lm}^M$ change slowly. We can then write the updates as:

$$\Delta v_{lm}^M = -\beta \frac{\partial E}{\partial q_K} Tag_{lm}^M = \beta \delta Tag_{lm}^M , \tag{16}$$

$$\Delta Tag_{lm}^M = -Tag_{lm}^M + \frac{\partial q_K}{\partial y_m^M} \frac{\partial y_m^M}{\partial a_m^M} \frac{\partial a_m^M}{\partial v_{lm}^M} , \tag{17}$$

$$= -Tag_{lm}^M + w_{Kj}^{'M} y_m^M(t)(1 - y_m^M(t)) \left[ \sum_{t'=0}^{t} x_l'(t') \right] . \tag{18}$$

Note that one can interpret a memory unit as a regular one that receives all sensory input in a trial simultaneously. For synapses onto memory units, we set $\lambda = 0$ to arrive at the last equation. The intuition behind the last equation is that because the activity of a memory unit does not decay, the influence of its inputs $x_l'$ on the activity in the motor layer does not decay either ($\lambda\gamma = 0$).

A special condition occurs when the environment returns the end-trial signal. In this case, the estimate $q_K$ in eqn. (6) is set to 0 (see [2]) and after the synaptic updates we reset the memory units and synaptic tags, so that there is no confounding between different trials.

AuGMEnT is biologically plausible because the information required for the synaptic updates is locally available by the interaction of feedforward and feedback signals and a globally released neuromodulator coding TD errors. As we will show, this mechanism is powerful enough to learn non-linear transformations and to create relevant working memories.

## 3 Experiments

We tested AuGMEnT on a set of memory tasks that have been used to investigate the effects of training on neuronal activity in area LIP. Across all of our simulations, we fixed the configuration of the association layer (three regular units, four memory units) and $Q$-layer (three output units, for directing gaze to the left, center or right of a virtual screen). The input layer was tailored to the specific task (see below). In all tasks, we trained the network by trial and error to fixate on a fixation mark and to respond to task-related cues. As is usual in training animals for complex tasks, we used a small shaping reward $r_{fix}$ (arbitrary units) to facilitate learning to fixate [17]. At the end of trials the model had to make an eye-movement to the left or right. The full task reward $r_{fin}$ was given if this saccade was accurate, while we aborted trials and gave no reward if the model made the wrong eye-movement or broke fixation before the go signal. We used a single set of parameters for the network; $\beta = 0.15$; $\lambda = 0.20$; $\gamma = 0.90$; $\epsilon = 0.025$ and $\theta = 2.5$, which shifts the sigmoidal activation function for association units so that that units with little input have almost zero output. Initial synaptic weights were drawn from a uniform distribution $U \sim [-0.25, 0.25]$. For all tasks we used $r_{fix} = 0.2$ and $r_{fin} = 1.5$.

### 3.1 Saccade/Antisaccade

The memory saccade/anti-saccade task (Fig. 2A) is based on [3]. This task requires a non-linear transformation and cannot be solved by a direct mapping from sensory units to $Q$-value units. Trials started with an empty screen, shown for one time step. Then either a black or white fixation mark was shown indicating a pro-saccade or anti-saccade trial, respectively. The model had to fixate on the fixation mark within ten time-steps, or the trial was terminated. After fixating for two time-steps, a cue was presented on the left or right and a small shaping reward $r_{fix}$ was given. The

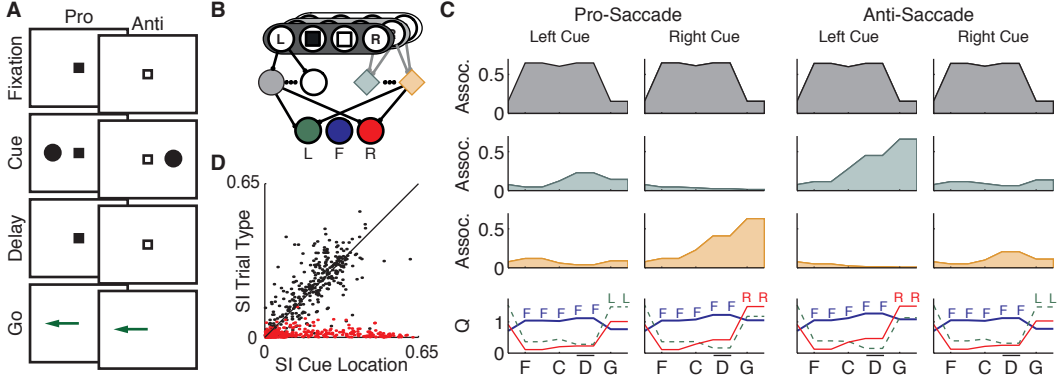

Figure 2: **A** Memory saccade/antisaccade task. **B** Model network. In the association layer, a regular unit and two memory units are color coded gray, green and orange, respectively. Output units $L$,$F$,$R$ are colored green, blue and red, respectively. **C** Unit activation traces for a sample trained network. Symbols in bottom graph indicate highest valued action. F, fixation onset; C, cue onset; D, delay; G, fixation offset ('Go' signal). Thick blue: fixate, dashed green: left, red: right. **D** Selectivity indices of memory units in saccade/antisaccade task (black) and in pro-saccade only task (red).

cue was shown for one time-step, and then only the fixation mark was visible for two time-steps before turning off. In the pro-saccade condition, the offset of the fixation mark indicated that the model should make an eye-movement towards the cue location to collect $r_{fin}$. In the anti-saccade condition, the model had to make an eye-movement away from the cue location. The model had to make the correct eye-movement within eight time steps. The input to the model (Fig. 2B) consisted of four binary variables representing the information on the virtual screen; two for the fixation marks and two for the cue location. Due to the $+/-$ cells, the input layer thus had 12 binary units.

We trained the models for at most $25,000$ trials, or until convergence. We measured convergence as the proportion of correct trials for the last 50 examples of all trial-types ($N = 4$). When this proportion reached $0.9$ or higher for all trial-types, learning in the network was stopped and we evaluated accuracy on all trial types without stochastic exploration of actions. We considered learning successful if the model performed all trial-types accurately.

We trained $10,000$ randomly initialized networks with and without a shaping reward ($r_{fix} = 0$). Of the networks that received fixation rewards, $9,945$ learned the task versus $7,641$ that did not receive fixation rewards; $\chi^2(1, N = 10,000) = 2,498, P < 10^{-6}$. The $10,000$ models trained with shaping learned the complete task in a median of $4,117$ trials. This is at least an order of magnitude faster than monkeys that typically learn such a task after months of training with more than $1,000$ trials per day, e. g. [6].

The activity of a trained network is illustrated in Fig. 2C. The $\mathcal{Q}$-unit for fixating at the center had strongest activity at fixation onset and throughout the fixation and memory delays, whereas the $\mathcal{Q}$-unit for the appropriate eye movement became more active after the go-signal. Interestingly, the activity of the $\mathcal{Q}$-cells also depended on cue-location during the memory delay, as is observed, for example, in the frontal eye fields [18]. This activity derives from memory units in the association layer that maintain a trace of the cue as persistent elevation of their activity and are also tuned to the difference between pro- and antisaccade trials. To illustrate this, we defined selectivity indices (SIs) to characterize the tuning of memory units to the difference between pro- or antisaccade trials and to the difference in cue location. The sensitivity of units to differences in trial types, $SI_{type}$ was $|0.5((R_{PL} + R_{PR}) - (R_{AL} + R_{AR}))|$, with $R$ representing a units' activation level (at 'Go' time) in pro (P) and anti-saccade trials (A) with a left (L) or right (R) cue. A unit has an SI of 0 if it does not distinguish between pro- and antisaccade trials, and an SI of 1 if it is fully active for one trial type and inactive for the other. The sensitivity to cue location, $SI_{cue}$, was defined $|0.5((R_{PL} + R_{AL}) - (R_{PR} + R_{AR}))|$. We trained 100 networks and found that units tuned to cue-location also tended to be selective for trial-type (black data points in Fig. 2D; SI correlation $0.79$, ($N = 400, P < 10^{-6}$)). To show that the association layer only learns to represent relevant features, we trained the same 100 networks using the same stimuli, but now only required pro-

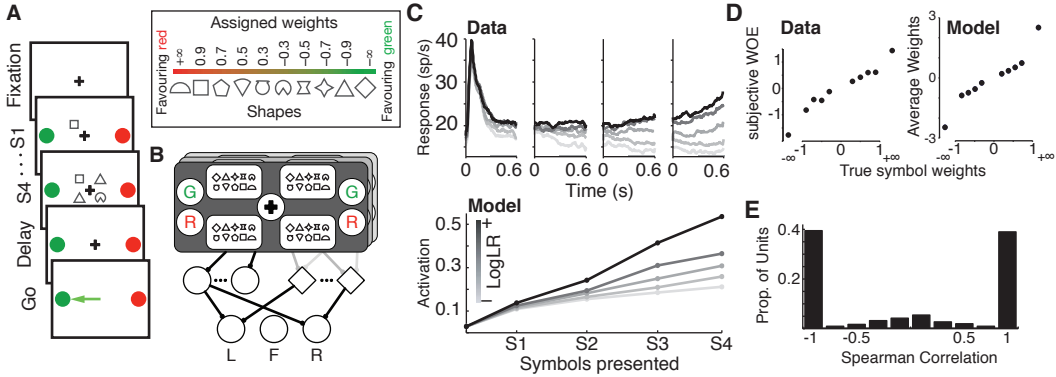

Figure 3: **A** Probabilistic classification task (redrawn from [6]). **B** Model network **C** Population averages, conditional on LogLR-quintile (*inset*) for LIP neurons (redrawn from [6]) (*top*) and model memory units over $100,000$ trials after learning had converged (*bottom*). **D** Subjective weights inferred for a trained monkey (redrawn from [6]) (*left*) and average synaptic weights to an example memory unit (*right*) versus true symbol weights (**A**, *right*). **E** Histogram of weight correlations for 400 memory units from 100 trained networks.

saccades, rendering the color of the fixation point irrelevant. Memory units in the 97 converged networks now became tuned to cue-location but not to fixation point color (Fig. 2D, red data points. SI Correlation 0.04, $(N = 388, P > 0.48)$), indicating that the association layer indeed only learns to represent relevant features.

## 3.2 Probabilistic Classification

Neurons in area LIP also play a role in perceptual decision making [5]. We hypothesized that memory units could learn to integrate probabilistic evidence for a decision. Yang and Shadlen [6] investigated how monkeys learn to combine information about four briefly presented symbols, which provided probabilistic cues whether a red or green eye movement target was baited with reward (Fig. 3A). A previous model with only one layer of modifiable synapses could learn a simplified, linear version of this task [19]. We tested if AuGMEnT could train the network to adapt to the full complexity of the task that demands a non-linear combination of information about the four symbols with the position of the red and green eye-movement targets. Trials followed the same structure as described in section 3.1, but now four cues were subsequently added to the display. Cues were drawn with replacement from a set of ten (Fig. 3A, right), each with a different associated weight. The sum of these weights, $W$, determined the probability that $r_{fin}$ was assigned to the red target $(R)$ as follows: $P(R|W) = 10^W/(1 + 10^W)$. For the green target $G$, $P(G|W) = 1 - P(R|W)$. At fixation mark offset, the model had to make a saccade to the target with the highest reward probability. The sensory layer of the model (Fig. 3B) had four retinotopic fields with binary units for all possible symbols, a binary unit for the fixation mark and four binary units coding the locations of the colored targets on the virtual screen. Due to the +/- units, this made $45 \times 3$ units in total.

As in [6], we increased the difficulty of the task gradually (i.e. we used a shaping strategy) by increasing the set of input symbols $(2, 4, \ldots, 10)$ and sequence length $(1 - 4)$ in eight steps. Training started with the 'trump' shapes which guarantee reward for the correct decision (Fig. 3A, right; see [6]) and then added the symbols with the next absolute highest weights. We determined that the task had been learned when the proportion of trials on which the correct decision was taken over the last $n$ trials reached $0.85$, where $n$ was increased with the difficulty level $l$ of the task. For the first 5 levels, $n(l) = 500 + 500l$ and for $l = 6, 7, 8$ $n$ was $10,000; 10,000$ and $20,000$, respectively. Networks were trained for at most $500,000$ trials.

The behavior of a trained network is shown in figure 3C (bottom). Memory units integrated information for one of the choices over the symbol sequence and maintained information about the value of this choice as persistent activity during the memory delay. Their activation was correlated to the log likelihood that the targets were baited, just like LIP neurons [6] (Fig. 3C). The graphs show average activations of populations of real and model neurons in the four cue presentation epochs. Each pos-

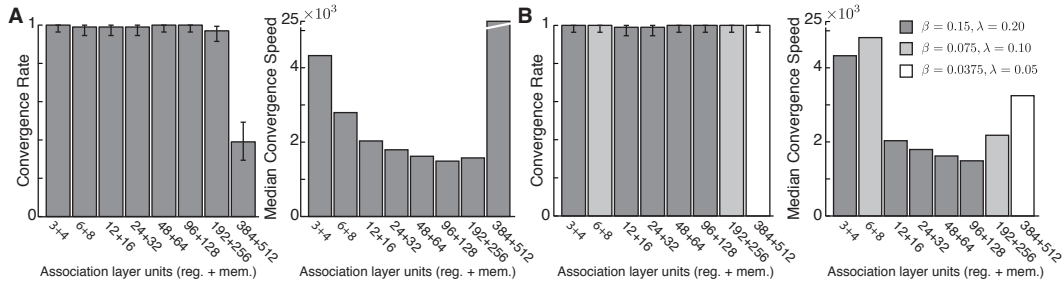

Figure 4: Association layer scaling behavior for **A** default learning parameters and, **B** optimized learning parameters. Error bars are $95\%$ confidence intervals. Parameters used are indicated by shading (see inset)

sible sub-sequence of cues was assigned to a log-likelihood ratio (logLR) quintile, which correlates with the probability that the neurons' preferred eye-movement is rewarded. Note that sub-sequences from the same trial might be assigned to different quintiles. We computed LogLR quintiles by enumerating all combinations of four symbols and then computing the probabilities of reward for saccades to red and green targets. Given these probabilities, we computed reward probability for all sub-sequences by marginalizing over the unknown symbols, i.e. to compute the probability that the red target was baited given only a first symbol $s_i$, $P(R|s_i)$, we summed the probabilities for full sequences starting with $s_i$ and divided by the number of such full sequences. We then computed the logLR for the sub-sequences and divided those into quintiles. For model units we rearranged the quintiles so that they were aligned in the last epoch to compute the population average.

Synaptic weights from input neurons to memory cells became strongly correlated to the true weights of the symbols (Fig. 3D, right; Spearman correlation, $\rho = 1, P < 10^{-6}$). Thus, the training of synaptic weights to memory neurons in parietal cortex can explain how the monkeys valuate the symbols [19]. We trained 100 networks on the same task and computed Spearman correlations for the memory unit weights with the true weights and found that in general they learn to represent the symbols (Fig. 3E). The learning scheme thus offers a biologically realistic explanation of how neurons in LIP learn to integrate relevant information in a probabilistic classification task.

### 3.3  Scaling behavior

To show that the learning scheme scales well, we ran a series of simulations with increasing numbers of association units. We scaled the number of association units by powers of two, from $2^1 = 2$ (yielding 6 regular units and 8 memory units) to $2^7 = 128$ (yielding 384 regular and 512 memory units). For each scale, we trained 100 networks on the saccade/antisaccade task, as described in section 3.1. We first evaluated these scaled networks with the standard set of learning parameters and found that these yielded stable results within a wide range but that performance deteriorated for the largest networks (from $2^6 = 64$; 192 regular units and 256 memory units) (Fig. 4A). In a second experiment (Fig. 4B), we also varied the learning rate ($\beta$) and trace decay ($\lambda$) parameters. We jointly scaled these parameters by $\frac{1}{2}, \frac{1}{4}$ and $\frac{1}{8}$ and selected the parameter combination which resulted in the highest convergence rate and the fastest median convergence speed. It can be seen that the performance of the larger networks was at least as good as that of the default network, provided the learning parameters were scaled. Furthermore, we ran extensive grid-searches over the $\lambda, \beta$ parameter space using default networks (not shown) and found that the model robustly learns both tasks with a wide range of parameters.

## 4  Discussion

We have shown that AuGMEnT can train networks to solve working memory tasks that require non-linear stimulus-response mappings and the integration of sensory evidence in a biologically plausible way. All the information required for the synaptic updates is available locally, at the synapses. The network is trained by a form of SARSA($\lambda$) [10, 2], and synaptic updates minimize TD errors by stochastic gradient descent. Although there is an ongoing debate whether SARSA or $\mathcal{Q}$-learning

[20] like algorithms are used by the brain [21, 22], we used SARSA because this has stronger convergence guarantees than $\mathcal{Q}$-learning when used to train neural networks [23]. Although stability is considered a problem for neural networks implementing reinforcement learning methods [24], AuGMEnT robustly trained networks on our tasks for a wide range of model parameters.

Technically, working memory tasks are Partially Observable Markov Decision Processes (POMDPs), because current observations do not contain the information to make optimal decisions [25]. Although AuGMEnT is not a solution for all POMDPs, as these are in general intractable [25], its simple learning mechanism is well able to learn challenging working memory tasks.

The problem of learning new working memory representations by reinforcement learning is not well-studied. Some early work used the biologically implausible backpropagation-through-time algorithm to learn memory representations [26, 27]. Most other work pre-wires some aspects of working memory and only has a single layer of plastic weights (e. g. [19]), so that the learning mechanism is not general. To our knowledge, the model by O'Reilly and Frank [7] is most closely related to AuGMEnT. This model is able to learn a variety of working memory tasks, but it requires a teaching signal that provides the correct actions on each time-step and the architecture and learning rules are elaborate. AuGMEnT only requires scalar rewards and the learning rules are simple and well-grounded in RL theory [2].

AuGMEnT explains how neurons become tuned to relevant sensory stimuli in sequential decision tasks that animals learn by trial and error. The scheme uses units with properties that resemble cortical and subcortical neurons: transient and sustained neurons in sensory cortices [28], action-value coding neurons in frontal cortex and basal ganglia [29, 30] and neurons which integrate input and therefore carry traces of previously presented stimuli in association cortex. The persistent activity of these memory cells could derive from intracellular processes, local circuit reverberations or recurrent activity in larger networks spanning cortex, thalamus and basal ganglia [31]. The learning scheme adopts previously proposed ideas that globally released neuromodulatory signals code deviations from reward expectancy and gate synaptic plasticity [8, 9, 14]. In addition to this neuromodulatory signal, plasticity in AuGMEnT is gated by an attentional feedback signal that tags synapses responsible for the chosen action. Such a feedback signal exists in the brain because neurons at the motor stage that code a selected action enhance the activity of upstream neurons that provided input for this action [32], a signal that explains a corresponding shift of visual attention [33]. AuGMEnT trains networks to direct feedback (i.e. selective attention) to features that are critical for the stimulus-response mapping and are associated with reward. Although the hypothesis that attentional feedback controls the formation of tags is new, there is ample evidence for the existence of synaptic tags [34, 12]. Recent studies have started to elucidate the identity of the tags [35, 36] and future work could investigate how they are influenced by attention. Interestingly, neuromodulatory signals influence synaptic plasticity even if released seconds or minutes later than the plasticity-inducing event [12, 35], which supports that they interact with a trace of the stimulus, i.e. some form of tag.

Here we have shown how interactions between synaptic tags and neuromodulatory signals explain how neurons in association areas acquire working memory representations for apparently disparate tasks that rely on working memory or decision making. These tasks now fit in a single, unified reinforcement learning framework.

## References

[1] Gnadt, J. and Andersen, R. A. Memory Related motor planning activity in posterior parietal cortex of macaque. *Experimental brain research.*, 70(1):216–220, 1988.

[2] Sutton, R. S. and Barto, A. G. *Reinforcement learning.* MIT Press, Cambridge, MA, 1998.

[3] Gottlieb, J. and Goldberg, M. E. Activity of neurons in the lateral intraparietal area of the monkey during an antisaccade task. *Nature neuroscience*, 2(10):906–12, 1999.

[4] Bisley, J. W. and Goldberg, M. E. Attention, intention, and priority in the parietal lobe. *Annual review of neuroscience*, 33:1–21, 2010.

[5] Gold, J. I. and Shadlen, M. N. The neural basis of decision making. *Annual review of neuroscience*, 30:535–74, 2007.

[6] Yang, T. and Shadlen, M. N. Probabilistic reasoning by neurons. *Nature*, 447(7148):1075–80, 2007.

[7] O'Reilly, R. C. and Frank, M. J. Making working memory work: a computational model of learning in the prefrontal cortex and basal ganglia. *Neural computation*, 18(2):283–328, 2006.

[8] Izhikevich, E. M. Solving the distal reward problem through linkage of STDP and dopamine signaling. *Cerebral cortex*, 17(10):2443–52, 2007.

[9] Montague, P. R., Hyman, S. E., et al. Computational roles for dopamine in behavioural control. *Nature*, 431(7010):760–7, 2004.

[10] Rummery, G. A. and Niranjan, M. Online Q-learning using connectionist systems. Technical report, Cambridge University Engineering Department, 1994.

[11] Funahashi, S., Bruce, C. J., et al. Mnemonic Coding of Visual Space in the Monkey's Dorsolateral Prefrontal Cortex. *Journal of Neurophysiology*, 6(2):331–349, 1989.

[12] Cassenaer, S. and Laurent, G. Conditional modulation of spike-timing- dependent plasticity for olfactory learning. *Nature*, 482(7383):47–52, 2012.

[13] Roelfsema, P. R. and van Ooyen, A. Attention-Gated Reinforcement Learning of Internal Representations for Classification. *Neural Computation*, 2214(17):2176–2214, 2005.

[14] Schultz, W. Multiple dopamine functions at different time courses. *Annual review of neuroscience*, 30:259–88, 2007.

[15] Wiering, M. and Schmidhuber, J. HQ-Learning. *Adaptive Behavior*, 6(2):219–246, 1997.

[16] Rumelhart, D. E., Hinton, G. E., et al. Learning representations by back-propagating errors. *Nature*, 323(6088):533–536, 1986.

[17] Krueger, K. A. and Dayan, P. Flexible shaping: how learning in small steps helps. *Cognition*, 110(3):380–94, 2009.

[18] Sommer, M. A. and Wurtz, R. H. Frontal Eye Field Sends Delay Activity Related to Movement, Memory, and Vision to the Superior Colliculus. *Journal of Neurophysiology*, 85(4):1673–1685, 2001.

[19] Soltani, A. and Wang, X.-J. Synaptic computation underlying probabilistic inference. *Nature Neuroscience*, 13(1):112–119, 2009.

[20] Watkins, C. J. and Dayan, P. Q-learning. *Machine learning*, 292:279–292, 1992.

[21] Morris, G., Nevet, A., et al. Midbrain dopamine neurons encode decisions for future action. *Nature neuroscience*, 9(8):1057–63, 2006.

[22] Roesch, M. R., Calu, D. J., et al. Dopamine neurons encode the better option in rats deciding between differently delayed or sized rewards. *Nature neuroscience*, 10(12):1615–24, 2007.

[23] van Seijen, H., van Hasselt, H., et al. A theoretical and empirical analysis of Expected Sarsa. *2009 IEEE Symposium on Adaptive Dynamic Programming and Reinforcement Learning*, pages 177–184, 2009.

[24] Baird, L. Residual algorithms: Reinforcement learning with function approximation. In *Proceedings of the 26th International Conference on Machine Learning (ICML)*, pages 30–37, 1995.

[25] Todd, M. T., Niv, Y., et al. Learning to use working memory in partially observable environments through dopaminic reinforcement. In *NIPS*, volume 21, pages 1689–1696, 2009.

[26] Zipser, D. Recurrent network model of the neural mechanism of short-term active memory. *Neural Computation*, 3(2):179–193, 1991.

[27] Moody, S. L., Wise, S. P., et al. A model that accounts for activity in primate frontal cortex during a delayed matching-to-sample task. *The journal of Neuroscience*, 18(1):399–410, 1998.

[28] Nassi, J. J. and Callaway, E. M. Parallel processing strategies of the primate visual system. *Nature reviews. Neuroscience*, 10(5):360–72, 2009.

[29] Hikosaka, O., Nakamura, K., et al. Basal ganglia orient eyes to reward. *Journal of neurophysiology*, 95(2):567–84, 2006.

[30] Samejima, K., Ueda, Y., et al. Representation of action-specific reward values in the striatum. *Science*, 310(5752):1337–40, 2005.

[31] Wang, X.-J. Synaptic reverberation underlying mnemonic persistent activity. *Trends in neurosciences*, 24(8):455–63, 2001.

[32] Roelfsema, P. R., van Ooyen, A., et al. Perceptual learning rules based on reinforcers and attention. *Trends in cognitive sciences*, 14(2):64–71, 2010.

[33] Deubel, H. and Schneider, W. Saccade target selection and object recognition: Evidence for a common attentional mechanism. *Vision Research*, 36(12):1827–1837, 1996.

[34] Frey, U. and Morris, R. Synaptic tagging and long-term potentiation. *Nature*, 385(6616):533–536, 1997.

[35] Moncada, D., Ballarini, F., et al. Identification of transmitter systems and learning tag molecules involved in behavioral tagging during memory formation. *PNAS*, 108(31):12931–6, 2011.

[36] Sajikumar, S. and Korte, M. Metaplasticity governs compartmentalization of synaptic tagging and capture through brain-derived neurotrophic factor (BDNF) and protein kinase Mzeta (PKMzeta). *PNAS*, 108(6):2551–6, 2011.

